# Generate Universal Adversarial Perturbations for Few-Shot Learning

**Yiman Hu**    **Yixiong Zou**∗    **Ruixuan Li**∗    **Yuhua Li**
School of Computer Science and Technology, Huazhong University of Science and Technology
`{imane, yixiongz, rxli, idcliyuhua}@hust.edu.cn`

## Abstract

Deep networks are known to be vulnerable to adversarial examples which are deliberately designed to mislead the trained model by introducing imperceptible perturbations to input samples. Compared to traditional perturbations crafted specifically for each data point, Universal Adversarial Perturbations (UAPs) are input-agnostic and shown to be more practical in the real world. However, UAPs are typically generated in a close-set scenario that shares the same classification task during the training and testing phases. This paper demonstrates the ineffectiveness of traditional UAPs in open-set scenarios like Few-Shot Learning (FSL). Through analysis, we identify two primary challenges that hinder the attacking process: the task shift and the semantic shift. To enhance the transferability of UAPs in FSL, we propose a unifying attacking framework addressing these two shifts. The task shift is addressed by aligning proxy tasks to the downstream tasks, while the semantic shift is handled by leveraging the generalizability of pre-trained encoders.The proposed Few-Shot Attacking FrameWork, denoted as FSAFW, can effectively generate UAPs across various FSL training paradigms and different downstream tasks. Our approach not only sets a new standard for state-of-the-art works but also significantly enhances attack performance, exceeding the baseline method by over 16%.

## 1 Introduction

Deep neural networks[16, 12] have made significant advancements in a variety of computer vision tasks. Nowadays, there is a growing trend of pre-training a model that achieves strong generalization capabilities and subsequently fine-tuning it for different unseen downstream tasks. A promising method to handle this open set problem is Few-Shot Learning (FSL), which learns a model that can rapidly adapt to unseen tasks with only a limited number of samples.

Meanwhile, deep networks have shown to be vulnerable to adversarial attacks [41, 14, 5], which are deliberately designed to deceive a trained model by introducing imperceptible perturbations to the input samples. Given the widespread adoption of the pre-training and fine-tuning paradigm, it is crucial to acknowledge and address the security concerns associated with such approaches.

For a more applicable way to attack various downstream tasks, we focus on Universal Adversarial Perturbations (UAPs) [26]. This kind of perturbation can be applied to all images once generated, eliminating the need for crafting image-dependent perturbations for each task.

However, current approaches for generating UAPs have primarily concentrated on close-set scenarios [26, 15, 30], where the classification tasks for both training and finetuning are essentially identical. Despite achieving a high Attack Success Rate (ASR) in their respective settings, our research reveals

---

∗Corresponding author.

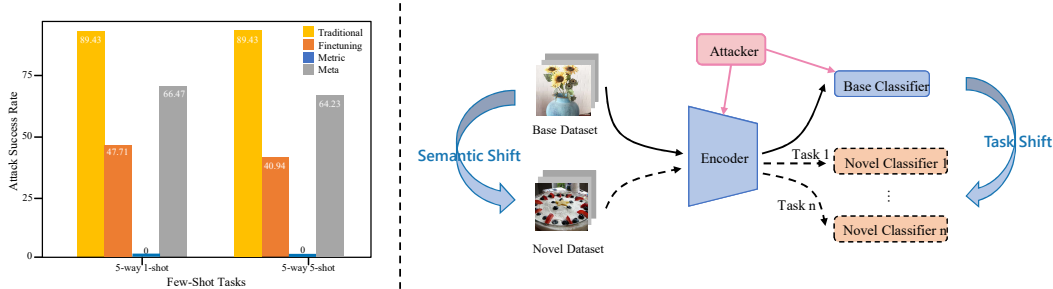

Figure 1: (Left) A decline in the Attack Success Rate (ASR) can be observed when conventional UAPs are applied to the FSL scenario. The yellow bar represents performance in traditional scenarios, while the other three indicate performance in different FSL training paradigms. For the metric-based paradigm, traditional methods even fail to generate UAPs. The detailed setting can be found in Section 4.1 (Right). Solid arrows represent the pre-training phase, while dotted arrows depict the testing phase. The attacker can only manipulate the pre-trained model, which suffers from the semantic shift of different datasets and the task shift from base to novel tasks.

that in FSL scenarios these techniques are ineffective, and in some cases, incapable of crafting UAPs, as shown in the left panel of Fig. 1.

To better understand how to effectively attack the FSL tasks, we conducted a comprehensive evaluation and analysis of the challenges involved, namely task shift and semantic shift, as depicted in the right panel of Fig. 1. To further fill up these two shifts and achieve a more generalizable UAP, we establish a baseline attacking framework as an initial point and progressively improve upon it. To address the task shift, we emphasize the significance of learning task bias for UAP, and handle this shift by constructing proxy tasks. To tackle the semantic shift, we demonstrate the effectiveness brought by the encoder's generalizability and eliminate the need for fine-tuning in the process of UAP generation.

By systematically addressing the shifts, we successfully generate the single perturbation only based on the parameters of the pre-trained encoder, without any prior knowledge of the pre-training dataset or downstream tasks. The ASRs of our universal perturbation surpass others by 16.49% for 5-way 1-shot tasks and 17.27% for 5-way 5-shot tasks. Additionally, our attacking framework(FSAFW) unifies the generation of UAPs in various FSL training paradigms, including finetuning-based [10, 47, 7], meta-based [13, 35, 39], and metric-based [43, 36, 50, 20] approaches, yielding state-of-the-art results in all cases. We summarize our main contributions as follows:

• We propose a new standard for the study of UAP in FSL scenarios, which highlights the limitations of traditional UAPs in the context of FSL.

• We provide a thorough analysis of the associated challenges, the task shift and semantic shift, which diminish the effectiveness of UAPs on downstream FSL tasks.

• We construct an attacking framework and fill up the two shifts step by step to enhance the attack performance, which is effective across various FSL paradigms.

• Our proposed method significantly advances state-of-the-art methods in FSL attacking performance, with an increase of over 16% in ASR in our standard.

## 2 Related Works

**Adversarial attacks** can be categorized into two types: image-dependent attacks and universal attacks. Image-dependent attacks have been widely studied [14, 23, 5, 22, 11]. The concept of Universal Adversarial Perturbations (UAPs) was initially introduced by [26] using an iterative Deepfool attack [27] applied to individual image samples. [28] developed a data-independent approach to generate UAPs. Furthermore, [49] validated the efficacy of using random source images. Several studies have explored the use of generative models to produce more generalized and natural-appearing UAPs. [15] were the first to utilize the generative network. [30] introduced GAP, a framework applicable to both classification and semantic segmentation models. [52] trained a generator by leveraging a

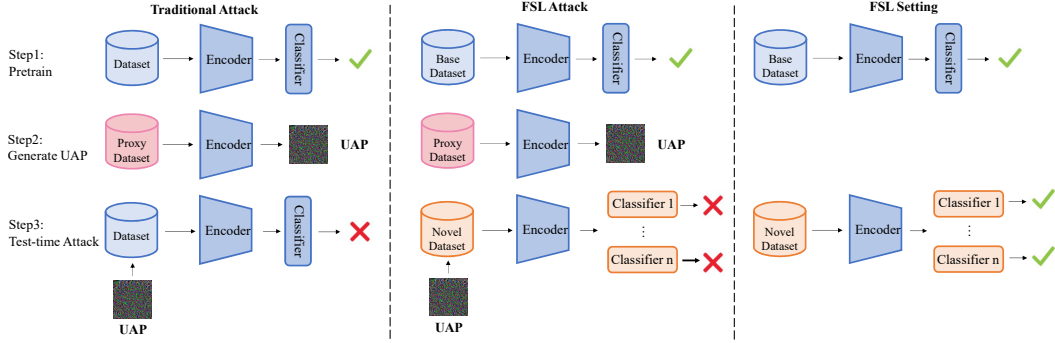

Figure 2: Comparison of different scenarios. In contrast to traditional settings, the FSL framework introduces two distinct challenges: the task shift from pre-training to testing, and the semantic shift from the base to the novel dataset. Attackers must overcome the two shifts to mount successful attacks in the FSL scenario.

contrastive loss function. However, none of these studies address the specific challenges associated with attacking few-shot learning tasks.

**Few-shot learning** is a machine learning paradigm that aims to recognize novel classes from a few labeled samples [25]. Existing FSL methods can be broadly grouped into three categories according to [21]: finetuning-based, meta-based, and metric-based methods. Finetuning-based strategies [47, 7, 10, 42, 32, 24] follow a transfer learning process that includes pre-training on base classes and fine-tuning on novel classes. Meta-based approaches [13, 33, 38, 35, 39, 3, 31] adopts a meta-learning paradigm to learn the cross-task knowledge through the optimization between the meta-learner and base-learner. In this way, the model adopts a quick adaptation to the novel dataset. Metric-based techniques [2, 40, 43, 36, 50, 20] focus on learning transferable representations and making predictions based on the distance between feature representations. This strategy eliminates the need for test-time fine-tuning. These paradigms have significantly advanced the progress of few-shot learning, yet a unified method for generating UAPs across all paradigms remains undeveloped.

## 3   UAP Setting for Few-Shot Learning

In this section, we outline the threat model of attacking the few-shot tasks and present key definitions and notations to facilitate a clearer understanding of the concepts involved.

### 3.1   Threat Model

We consider an attacker who aims to craft a universal adversarial perturbation to attack a pre-trained victim model and further impair the performance of downstream few-shot tasks. The attacker only has access to the pre-trained model (*e.g.*, by downloading from public repositories), but can not obtain the datasets used for pre-training and have no knowledge of the following few-shot tasks. Once the perturbation is generated, the attacker attaches it to each query sample. The crafted perturbation is imperceptible, and is expected to greatly mislead the few-shot classification.

### 3.2   Definition and Notations

In the domain of few-shot learning, abundant annotated images of the base dataset $\mathcal{D}_b$ can be used for pre-training. Subsequently, the model would be fine-tuned using limited samples from the novel dataset $\mathcal{D}_n$, where the categories in $\mathcal{D}_n$ do not overlap with those in $\mathcal{D}_b$. In the pre-training stage, the victim model $f(\cdot)$ that composed of an encoder $f_e(\cdot)$ and a base classifier $f_{bc}(\cdot)$ is trained on $\mathcal{D}_b$. In the evaluation stage, different forms of tasks are sampled from $\mathcal{D}_n$. Each task contains a support set $\mathcal{S}$ used for finetuning and a query set $\mathcal{Q}$ used for evaluation. The support set includes $n$ different classes with $k$ samples per class, referred to as $n$-way $k$-shot. Similarly, the query set contains $n$ classes with $q$ samples per class. For each task, a novel class classifier $f_{nc}(\cdot)$ is newly trained on $\mathcal{S}$. For any image $x$ in $\mathcal{Q}$, $f_{nc}(f_e(x), \mathcal{S})$ yields the classifier's output. The goal of FSL is to learn a

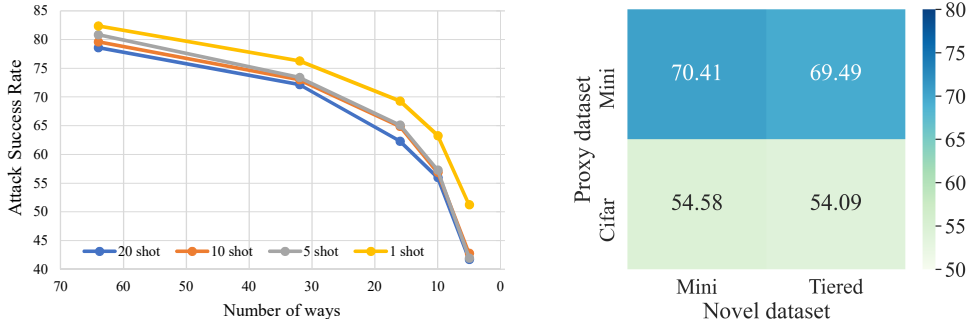

(a) An illustration of the ASRs under different downstream tasks.

(b) An illustration of the ASRs under different datasets.

Figure 3: The ASRs under different task shifts and semantic shifts. In (b), Cifar, Mini, Tiered are different datasets detailed in Section 6.1.

generalizable $f_e(\cdot)$ that exhibits good performance on $\mathcal{Q}$ across various tasks after learning from the limited examples in $\mathcal{S}$.

Once get the pre-trained encoder $f_e(\cdot)$, an attacker can craft the adversarial images as follows: $x^{adv} = x + \delta$, where $\delta$ represents the perturbation. To ensure that the adversarial images are visually indistinguishable from the original ones, $\delta$ is usually restricted by a chosen distance metric, such as the $\ell_\infty$ norm. This constraint can be expressed as $\|x^{adv} - x\|_\infty = \|\delta\|_\infty \leq \epsilon$. An effective approach to generate $\delta$ is to train a generator $g_\theta(\cdot)$ on the proxy dataset $\mathcal{D}_p$. The generator takes a random vector $z$ as input and it aims to produce perturbations that satisfy the given constraints while maximizing adversarial impact.

## 4   A Closer Look at UAP Generation in Few-Shot Learning

In this session, we conduct a thorough analysis of the challenges associated with generating Universal Adversarial Perturbations (UAPs) in the context of Few-Shot Learning (FSL). We first compare the attack performance in the traditional scenario with that in the FSL scenario. Then we point out the lower performance in the FSL scenario is due to the presence of two shifts.

### 4.1   Poor Performance of Traditional UAPs in the FSL Scenario

To better understand the performance of existing attacking methods on FSL tasks, we directly apply the traditional method in the FSL scenario. We compare the Attack Success Rate (ASR) in the FSL scenario with that observed in the original scenario. The ASR refers to the success rate of the UAP in fooling the classifier. A higher ASR means a stronger attack capability.

We adopt the *Generative Adversarial Perturbation (GAP)*[30] method, due to its straightforward applicability to FSL tasks and its high attack performance within its original context. The GAP method generates UAP through a ResNet Generator[17], denoted as $g_{GAP}$. The perturbation $\delta$ is produced as follows:

$$\delta = g_{GAP}(z), \tag{1}$$

where $z$ represents a random vector. The generator is optimized with respect to the following loss term:

$$L_{GAP} = -\log(\mathcal{H}(f(x + \delta), y)). \tag{2}$$

Here, $(x, y)$ is the data point sampled from proxy dataset, $f(\cdot)$ denotes the model pre-trained on the base dataset, and $\mathcal{H}(f(x^{adv}), y)$ represents the cross-entropy loss.

For the traditional GAP setting, we compute the mean ASR on VGG16, VGG19, and ResNet152 reported in the original paper to gauge the average performance in the traditional scenario. In the FSL scenario, where the attacker does not have access to the base or novel datasets, we utilize CIFAR-FS [4] as the proxy dataset to generate UAPs. We attack three pre-trained FSL models representative of different paradigms: finetuning-based, metric-based, and meta-based approaches.

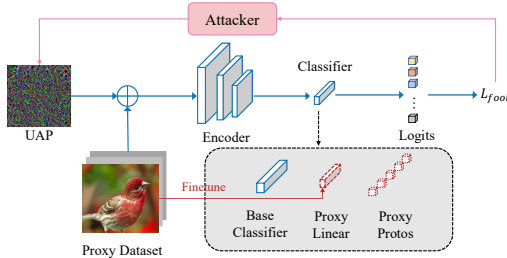

(a) An illustration of the attacking framework.

| Methods | TS | SS | ASR | |
|---|---|---|---|---|
| | | | 1-shot | 5-shot |
| Base classifier | ✗ | ✗ | $65.07_{\pm 0.36}$ | $61.73_{\pm 0.30}$ |
| Proxy linear | ✓ | ✗ | $70.87_{\pm 0.36}$ | $68.87_{\pm 0.27}$ |
| Base linear | ✓ | ✓ | $76.75_{\pm 0.32}$ | $74.67_{\pm 0.21}$ |
| Proxy protos | ✓ | ✓ | $\mathbf{80.04_{\pm 0.30}}$ | $\mathbf{77.69_{\pm 0.21}}$ |

(b) An illustration of different ASRs.

Figure 4: (a) The Base Classifier is the pre-trained classifier. Proxy Linear and Proxy Protos are newly designed to address the task shift and the semantic shift, as detailed in Section 5. (b) The detailed Attack Success Rate(ASR) of different methods on 5-way 1-shot and 5-way 5-shot tasks. TS, SS represents whether the Task Shift and the Semantic Shift are handled respectively.

As illustrated in the left panel of figure 1, GAP achieves a pretty high mean ASR of 89.43% in traditional tasks. However, when switched to the FSL setting, there is a marked decrease in performance, with a reduction of at least 25% observed in both 5-way 1-shot and 5-way 5-shot tasks. Moreover, the GAP method is unable to generate UAPs for metric-based models, as its generation process depends on the presence of a fixed classifier, which is lacking for metric-based methods. This observation leads us to pose the question: **What makes the attack success rate of UAPs decrease so much in FSL?** In the next subsection, we will discuss the factors.

## 4.2 Two Shifts in FSL Affect the Attack Transferability

We compare the differences in the UAP generation process between the traditional scenario and the FSL scenario, as depicted in Figure 2. In traditional scenarios, UAP generation relies on a fixed classifier that remains unchanged throughout both the training and testing phases. In contrast, FSL scenarios present a variety of classification tasks during testing—encompassing different categories and shapes, such as 5-way 1-shot and 5-way 5-shot, thereby posing challenges to UAP generalization. Therefore, our first hypothesis is that **the task shift in FSL hinders the transferability of the UAP**.

To validate our hypothesis, we conducted experiments to evaluate the performance of the UAP across different downstream tasks. The experiments vary the number of ways from 5 to 64 and the number of shots from 1 to 20. To maintain the semantic consistency between the pre-training and testing datasets, we sampled an additional 100 images for each category in the training split of *mini*-ImageNet to serve as the downstream dataset. The results are presented in Figure 3 (a), demonstrating that as the downstream task becomes less similar to the pre-training task(64-way full shot), the attacking performance of the UAP diminishes.

From Figure 2, it can also be observed that the categories in training and testing datasets do not overlap in the FSL scenario. This semantic shift may contribute to a reduction in the attacking performance. Therefore, we derive a second hypothesis that **the semantic shift in FSL hinders the transferability of the UAP**. To verify the impact of the semantic shift, we keep the downstream task constant and vary the proxy dataset and downstream dataset. We choose a model pre-trained on the *mini*-ImageNet as the victim model. As illustrated in Figure 3 (b), the attack performance declines with increasing distances between both the downstream dataset and the pre-trained dataset(mini), as well as between the proxy dataset and the pre-trained dataset(mini).

To summarize, we empirically demonstrate that the two shifts that existed in the FSL scenario degrade the attack performance of traditionally generated UAPs.

## 5 Method

In this section, we propose a baseline framework aimed at generating effective Universal Adversarial Perturbations (UAPs) within a Few-Shot Learning (FSL) scenario. Starting with an initial attacking framework, we progressively refine our approach to address the two shifts mentioned before. Finally, we have developed a universally applicable UAP generation strategy, capable of attacking models

trained under all kinds of FSL paradigms and showing strong generalizability to different downstream tasks.

## 5.1 A Baseline Attacking Framework

The attacking framework is illustrated in Figure 4 (a). The victim model, built on a ResNet12[16] backbone, is pre-trained using the training split of the *mini*-ImageNet[43] dataset, following the training paradigm established in [10]. For the attack, we train a generator to transform random vectors into a UAP, which is also used in [52]. Since the attacker lacks access to both the training and testing data from the *mini*-ImageNet dataset, we opt for the training split of the CIFAR-FS dataset[4] as a proxy. To optimize our generator $g_\theta$, we employ a negative cross-entropy loss as the attacking objective, defined by the following equation:

$$L_{fool} = -\mathcal{H}(f_{bc}(f_e(x + g_\theta(z))), y). \tag{3}$$

As the original approach, we utilize a 64-way classifier $f_{bc}(\cdot)$ from the baseline model to infer, without any fine-tuning. The results are presented in the first row of Figure 4 (b), denoted as **base classifier**. The results are far from satisfactory compared to the performance in traditional tasks, highlighting the need for further refinement in the FSL scenario.

## 5.2 Fill Up the Task Shift through Task Alignment

As demonstrated in Section 4.2, a misalignment between upstream and downstream tasks can adversely affect the performance of UAPs. Given that the attacker cannot foresee the specific downstream tasks, we modify the tasks used in UAP generation to better align with those downstream tasks. Importantly, we only aim to approximate the shape of the downstream tasks rather than achieving exactness, which is sufficient to let the UAP acquire the necessary task bias. This is supported by the ablation study detailed in Section 6.3. We sample *proxy tasks* with the shape of 5-way 1-shot from the proxy dataset $\mathcal{D}_p$. For each task, we construct a linear classifier $f_{pc}(\cdot)$ based on its support set $\mathcal{S}_p$. We refer to this method as **proxy linear**. The objective function for training the classifier is given by:

$$L_{pc} = \mathcal{H}(f_{pc}(f_e(x)), y), \tag{4}$$

where $(x, y)$ represents the data sampled from $\mathcal{S}_p$. Once the proxy linear classifier has been trained, the attacker generates perturbations guided by the perturbed query set $\mathcal{Q}_p^{adv}$ from the proxy task, as depicted in Figure 4 (a). The fooling loss is quite like equation 3 except that the classifier is newly constructed and the data $(x, y)$ is sampled from $\mathcal{Q}_p$, rather than the entire proxy dataset. The fooling loss is calculated as follows to optimize the generator $g_\theta$ :

$$L_{fool} = -\mathcal{H}(f_{pc}(f_e(x + g_\theta(z))), y), \tag{5}$$

where $(x, y)$ is sampled from $\mathcal{Q}_p$.

By integrating task alignment into the training of UAPs, we empower them to generalize more effectively to downstream tasks, even in the absence of direct access to those tasks. From the results presented in the second row of Figure 4 (b), it is evident that by addressing the task shift, there is a marked improvement in the ASR, with an increase of 5% for 1-shot tasks and 7% for 5-shot tasks.

## 5.3 Fill Up the Semantic Shift by Leveraging the Encoder's Generalizability

**Analysis of the semantic shift.** In Section 4.2, we discussed the impact of the semantic shift on UAPs, a phenomenon widely present in FSL. As attackers lack access to upstream or downstream datasets, our strategy emphasizes utilizing the generalization capabilities of the pre-trained encoder. A well-trained encoder possesses the capability to transfer useful information to the novel dataset, even trained on the base dataset. Consequently, we hypothesize that *if the proxy dataset closely aligns with the base dataset, the generated UAP can be more transferable to the novel dataset.*

We substitute the proxy dataset with the base dataset to create an ideal scenario for testing our assumption. While attackers cannot access the base dataset in real-world situations, utilizing it in our experiments allows us to better understand and validate our hypotheses. To be specific, we sample different tasks from the base dataset and train a linear classifier for each task, following the process outlined in equation 4. The results are documented in the third row of Figure 4 (b), denoted as **base**

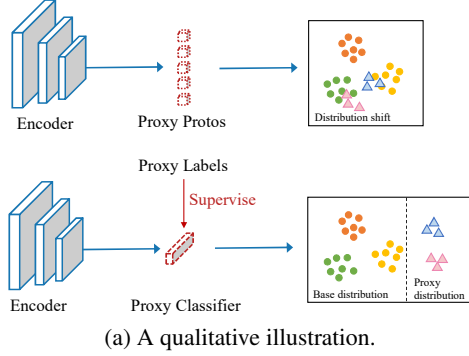

| | Log | LS | ASR | |
|---|---|---|---|---|
| | | | 1-shot | 5-shot |
| eq 7 | ✗ | ✗ | $80.04_{\pm 0.30}$ | $77.69_{\pm 0.21}$ |
| eq 8 | ✓ | ✗ | $80.72_{\pm 0.30}$ | $78.35_{\pm 0.21}$ |
| eq 9 | ✓ | ✓ | $\mathbf{81.56_{\pm 0.29}}$ | $\mathbf{79.00_{\pm 0.18}}$ |

(a) A qualitative illustration.     (b) An illustration of different ASRs.

Figure 5: (a) A qualitative illustration showing the effect of the supervisory signal. (b) The attack performance comparison of different fooling losses.

**linear**. Compared to the Proxy liner, a further improvement can be observed in the table. The results suggest that a closer simulation of the base dataset distribution enhances the transferability of the UAP to the novel dataset.

**Alleviation of the semantic shift.** While utilizing the base dataset proves effective, the attacker can only utilize the proxy dataset. As discussed in [46], when the proxy dataset is fed into the encoder, its output distribution tends to shift towards the base dataset, which is exactly what we need.

However, the previously mentioned proxy linear method introduces a supervisory signal that corrects the distribution, as shown in Figure 5 (a). The output distribution, initially altered by the encoder, is adjusted by the constructed proxy classifier.

This causes the UAP to lose its connection with the base dataset during training, thereby diminishing its ability to generalize to downstream tasks. Consequently, we abandon the proxy linear classifier and adopt a metric-based evaluation to mitigate the influence of labels from the proxy dataset. For each task sampled from the proxy dataset, we extract the output features from the support set and compute the mean of each class (i.e., class prototypes). The prototype for class $k$ in support set $\mathcal{S}_p$ is calculated using the equation:

$$c_k = \frac{1}{N_k} \sum_{y=k} f_e(x). \tag{6}$$

$(x, y)$ is sampled from $\mathcal{S}_p$ and $N_k$ represents the number of samples for class $k$ in the support set. We then employ a distance metric $D(\cdot, \cdot)$ to quantify the distance between the query feature and each class prototype. The classification probability for a query sample $x$ in class $k$ is then determined by:

$$p(y = k|x) = \frac{e^{D(f_e(x), c_k)}}{\sum_{k'} e^{D(f_e(x), c_{k'})}}, \tag{7}$$

where $k'$ represents a class in the task. In accordance with prior work [36], we utilize cosine similarity as the distance metric.

The **proxy protos** strategy facilitates a closer approximation to the novel dataset, regardless of the specific proxy dataset used. The efficacy of this approach is documented in the final row of Figure 4 (b), which presents the most favorable results among all methods tested.

**A further improvement.** As suggested by [30], an alternative formulation for the fooling loss is as follows:

$$L_{fool} = -\log \mathcal{H}(f_{proto}(f_e(x + g(z))), y), \tag{8}$$

where $f_{proto}$ denotes the predicted classification computed using the prototype-based approach previously described. Furthermore, to avoid reliance on the proxy query label and enhance transferability, we can supervise the perturbation generator with smoothed labels, denoted as $\hat{y}$. The fooling loss is redefined as:

$$L_{fool} = -\log \mathcal{H}(f_{proto}(f_e(x + g(z))), \hat{y}). \tag{9}$$

Table 1: 5-way 1-shot ASR results of our framework in different FSL victim models.

| Victim | Proxy | Backbone | Baseline | Baseline++ | ANIL-1 | R2D2-1 | ProtoNet | DN4 |
|---|---|---|---|---|---|---|---|---|
| Mini | Cifar | RN12 | $81.56_{\pm0.29}$ | $58.94_{\pm0.43}$ | $77.84_{\pm0.28}$ | $70.34_{\pm0.29}$ | $69.03_{\pm0.36}$ | $73.31_{\pm0.32}$ |
| | | RN18 | $71.55_{\pm0.35}$ | $74.67_{\pm0.28}$ | $72.11_{\pm0.29}$ | $70.00_{\pm0.31}$ | $60.02_{\pm0.34}$ | $67.66_{\pm0.30}$ |
| | Mini | RN12 | $81.63_{\pm0.28}$ | $68.55_{\pm0.46}$ | $78.08_{\pm0.28}$ | $77.22_{\pm0.28}$ | $68.54_{\pm0.36}$ | $74.91_{\pm0.33}$ |
| | | RN18 | $72.32_{\pm0.36}$ | $80.01_{\pm0.26}$ | $76.87_{\pm0.30}$ | $77.42_{\pm0.30}$ | $76.09_{\pm0.31}$ | $69.36_{\pm0.32}$ |
| | Tiered | RN12 | $79.68_{\pm0.30}$ | $67.51_{\pm0.43}$ | $77.81_{\pm0.28}$ | $77.09_{\pm0.27}$ | $72.87_{\pm0.34}$ | $75.23_{\pm0.33}$ |
| | | RN18 | $72.11_{\pm0.36}$ | $79.45_{\pm0.27}$ | $76.36_{\pm0.30}$ | $77.11_{\pm0.30}$ | $74.80_{\pm0.32}$ | $73.31_{\pm0.31}$ |
| Tiered | Cifar | RN12 | $76.03_{\pm0.32}$ | $62.56_{\pm0.29}$ | $68.26_{\pm0.28}$ | $76.49_{\pm0.27}$ | $76.73_{\pm0.32}$ | $78.47_{\pm0.34}$ |
| | | RN18 | $69.27_{\pm0.32}$ | $73.82_{\pm0.27}$ | $69.27_{\pm0.29}$ | $70.31_{\pm0.30}$ | $67.60_{\pm0.31}$ | $63.66_{\pm0.39}$ |
| | Mini | RN12 | $78.91_{\pm0.30}$ | $75.67_{\pm0.29}$ | $75.02_{\pm0.31}$ | $77.68_{\pm0.28}$ | $76.71_{\pm0.32}$ | $78.80_{\pm0.34}$ |
| | | RN18 | $74.74_{\pm0.33}$ | $80.03_{\pm0.24}$ | $75.68_{\pm0.32}$ | $78.08_{\pm0.29}$ | $71.65_{\pm0.33}$ | $70.58_{\pm0.43}$ |
| | Tiered | RN12 | $78.84_{\pm0.30}$ | $75.99_{\pm0.28}$ | $75.48_{\pm0.31}$ | $77.64_{\pm0.28}$ | $76.83_{\pm0.32}$ | $78.94_{\pm0.35}$ |
| | | RN18 | $73.51_{\pm0.32}$ | $80.44_{\pm0.25}$ | $75.33_{\pm0.33}$ | $78.25_{\pm0.28}$ | $71.45_{\pm0.34}$ | $69.15_{\pm0.43}$ |

Table 2: 5-way 5-shot ASR results of our framework in different FSL victim models.

| Victim | Proxy | Backbone | Baseline | Baseline++ | ANIL-1 | R2D2-1 | ProtoNet | DN4 |
|---|---|---|---|---|---|---|---|---|
| Mini | Cifar | RN12 | $79.00_{\pm0.18}$ | $63.41_{\pm0.27}$ | $78.31_{\pm0.21}$ | $70.42_{\pm0.22}$ | $67.96_{\pm0.28}$ | $74.88_{\pm0.22}$ |
| | | RN18 | $71.82_{\pm0.21}$ | $73.87_{\pm0.24}$ | $71.37_{\pm0.25}$ | $69.74_{\pm0.23}$ | $59.94_{\pm0.31}$ | $69.75_{\pm0.22}$ |
| | Mini | RN12 | $79.05_{\pm0.19}$ | $75.01_{\pm0.22}$ | $78.40_{\pm0.21}$ | $79.14_{\pm0.18}$ | $70.09_{\pm0.27}$ | $76.90_{\pm0.20}$ |
| | | RN18 | $73.89_{\pm0.21}$ | $79.31_{\pm0.20}$ | $77.46_{\pm0.21}$ | $79.09_{\pm0.17}$ | $76.34_{\pm0.19}$ | $68.70_{\pm0.24}$ |
| | Tiered | RN12 | $76.78_{\pm0.21}$ | $74.03_{\pm0.23}$ | $78.43_{\pm0.20}$ | $79.02_{\pm0.18}$ | $74.26_{\pm0.22}$ | $76.91_{\pm0.21}$ |
| | | RN18 | $73.14_{\pm0.23}$ | $78.99_{\pm0.20}$ | $77.42_{\pm0.21}$ | $78.92_{\pm0.17}$ | $74.75_{\pm0.21}$ | $75.93_{\pm0.21}$ |
| Tiered | Cifar | RN12 | $75.09_{\pm0.21}$ | $59.40_{\pm0.30}$ | $68.96_{\pm0.23}$ | $76.94_{\pm0.19}$ | $78.33_{\pm0.17}$ | $75.91_{\pm0.21}$ |
| | | RN18 | $70.32_{\pm0.24}$ | $71.30_{\pm0.25}$ | $70.73_{\pm0.23}$ | $71.55_{\pm0.22}$ | $70.87_{\pm0.22}$ | $66.09_{\pm0.28}$ |
| | Mini | RN12 | $78.43_{\pm0.17}$ | $78.07_{\pm0.17}$ | $76.05_{\pm0.22}$ | $78.72_{\pm0.16}$ | $78.34_{\pm0.17}$ | $76.42_{\pm0.22}$ |
| | | RN18 | $78.01_{\pm0.16}$ | $79.41_{\pm0.18}$ | $77.90_{\pm0.21}$ | $79.25_{\pm0.15}$ | $76.65_{\pm0.17}$ | $74.85_{\pm0.26}$ |
| | Tiered | RN12 | $78.48_{\pm0.16}$ | $78.09_{\pm0.18}$ | $76.61_{\pm0.22}$ | $78.79_{\pm0.16}$ | $78.37_{\pm0.17}$ | $77.09_{\pm0.23}$ |
| | | RN18 | $76.60_{\pm0.18}$ | $79.83_{\pm0.18}$ | $77.88_{\pm0.21}$ | $79.14_{\pm0.16}$ | $75.43_{\pm0.19}$ | $73.06_{\pm0.27}$ |

The adoption of this revised fooling loss yields further performance enhancements, as evidenced in Figure 5 (b). Consequently, we have adopted equation 9 as the final formulation for our fooling loss.

## 5.4 Conclusion and Discussion

Through a thorough comparison of the traditional scenario and the FSL scenario, we point out two critical shifts that reduce the attack performance on FSL tasks. To address the task shift, we sample proxy tasks to mirror the shape of the downstream tasks. To handle the semantic shift, we employ class prototypes to train UAPs, circumventing the introduction of the proxy labels. Moreover, the application of smoothed labels yields additional enhancements in attack performance.

Our final method can be summarized in the following steps: *(1)* Sample 5-way 1-shot tasks from the proxy dataset. *(2)* Calculate prototypes for each class in the proxy tasks. *(3)* Compute the fooling loss based on the distance between the query sample and class prototypes. *(4)* Optimize the perturbation generator to produce a highly transferable UAP in FSL. By following these steps, our method systematically generates universal adversarial perturbations that are robust and generalizable across different tasks.

## 6 Experiments

### 6.1 Implementation Details

**Datasets.** We utilize three widely used datasets in Few-Shot Learning (FSL) as the proxy dataset: CIFAR-FS[4], *mini*-ImageNet[43], and Tiered-ImageNet[34]. Both CIFAR-FS and *mini*-ImageNet

Table 3: Comparison of different attack methods on ASR for 5-way 1-shot tasks.

| Victim | Method | Mark | Baseline | Baseline++ | ANIL-1 | R2D2-1 | ProtoNet | DN4 |
|---|---|---|---|---|---|---|---|---|
| Mini | UAN | SPW-18 | $52.27_{\pm 0.33}$ | $47.68_{\pm 0.42}$ | $43.64_{\pm 0.26}$ | - | - | - |
| | GAP | CVPR-18 | $47.71_{\pm 0.31}$ | $49.40_{\pm 0.35}$ | $66.47_{\pm 0.32}$ | - | - | - |
| | AdvEncoder | ICCV-23 | $76.68_{\pm 0.31}$ | $57.37_{\pm 0.38}$ | $68.51_{\pm 0.31}$ | $59.10_{\pm 0.29}$ | $66.63_{\pm 0.34}$ | $72.85_{\pm 0.31}$ |
| | FSAFW | Ours | $\mathbf{81.56_{\pm 0.29}}$ | $\mathbf{58.94_{\pm 0.43}}$ | $\mathbf{77.84_{\pm 0.28}}$ | $\mathbf{70.34_{\pm 0.29}}$ | $\mathbf{69.03_{\pm 0.36}}$ | $\mathbf{73.31_{\pm 0.32}}$ |
| Tiered | UAN | SPW-18 | $40.34_{\pm 0.31}$ | $33.00_{\pm 0.27}$ | $51.91_{\pm 0.28}$ | - | - | - |
| | GAP | CVPR-18 | $49.72_{\pm 0.33}$ | $58.23_{\pm 0.28}$ | $61.19_{\pm 0.30}$ | - | - | - |
| | AdvEncoder | ICCV-23 | $75.99_{\pm 0.32}$ | $62.16_{\pm 0.29}$ | $53.82_{\pm 0.30}$ | $71.01_{\pm 0.31}$ | $60.23_{\pm 0.33}$ | $68.86_{\pm 0.32}$ |
| | FSAFW | Ours | $\mathbf{76.03_{\pm 0.32}}$ | $\mathbf{62.56_{\pm 0.29}}$ | $\mathbf{68.26_{\pm 0.28}}$ | $\mathbf{76.49_{\pm 0.27}}$ | $\mathbf{76.73_{\pm 0.32}}$ | $\mathbf{78.47_{\pm 0.34}}$ |

Table 4: Comparison of different attack methods on ASR for 5-way 5-shot tasks.

| Victim | Method | Mark | Baseline | Baseline++ | ANIL-1 | R2D2-1 | ProtoNet | DN4 |
|---|---|---|---|---|---|---|---|---|
| Mini | UAN | SPW-18 | $46.09_{\pm 0.30}$ | $45.42_{\pm 0.30}$ | $42.23_{\pm 0.26}$ | - | - | - |
| | GAP | CVPR-18 | $40.94_{\pm 0.28}$ | $45.71_{\pm 0.29}$ | $64.23_{\pm 0.29}$ | - | - | - |
| | AdvEncoder | ICCV-23 | $74.19_{\pm 0.23}$ | $55.12_{\pm 0.28}$ | $67.34_{\pm 0.26}$ | $59.47_{\pm 0.25}$ | $67.76_{\pm 0.26}$ | $74.72_{\pm 0.21}$ |
| | FSAFW | Ours | $\mathbf{79.00_{\pm 0.18}}$ | $\mathbf{63.41_{\pm 0.27}}$ | $\mathbf{78.31_{\pm 0.21}}$ | $\mathbf{70.42_{\pm 0.22}}$ | $\mathbf{67.96_{\pm 0.28}}$ | $\mathbf{74.88_{\pm 0.22}}$ |
| Tiered | UAN | SPW-18 | $32.97_{\pm 0.29}$ | $23.45_{\pm 0.23}$ | $51.75_{\pm 0.27}$ | - | - | - |
| | GAP | CVPR-18 | $44.90_{\pm 0.30}$ | $52.40_{\pm 0.28}$ | $59.52_{\pm 0.28}$ | - | - | - |
| | AdvEncoder | ICCV-23 | $75.03_{\pm 0.23}$ | $58.23_{\pm 0.28}$ | $53.25_{\pm 0.28}$ | $70.93_{\pm 0.25}$ | $60.40_{\pm 0.30}$ | $69.55_{\pm 0.22}$ |
| | FSAFW | Ours | $\mathbf{75.09_{\pm 0.21}}$ | $\mathbf{59.40_{\pm 0.30}}$ | $\mathbf{68.96_{\pm 0.23}}$ | $\mathbf{76.94_{\pm 0.19}}$ | $\mathbf{78.33_{\pm 0.17}}$ | $\mathbf{75.91_{\pm 0.21}}$ |

datasets comprise images from 100 categories, divided into 64 classes for training, 16 for validation, and 20 for testing. Tiered-ImageNet, a more extensive subset of ImageNet [9], consists of 608 classes (779,165 images) organized into 34 high-level categories, which are further split into 351 training classes, 97 validation classes, and 160 testing classes. To prevent the attacker from gaining knowledge of the downstream images, only the training portions of these datasets are utilized for the generation of UAPs.

**Training Details.** All the victim models are downloaded from the Libfewshot[21], a comprehensive Library for FSL. In line with the library's classification of FSL training paradigms, we selected Baseline and Baseline++ [10] as representatives of the finetuning-based paradigm, ANIL [31] and R2D2 [3] for the meta-based paradigm, and ProtoNet [36] along with DN4 [20] for the metric-based paradigm. For each paradigm, we adopted ResNet12 and ResNet18[16] as the backbone of the victim models, following the previous works[8, 48]. All these victim models are pre-trained on the *mini*-ImageNet and Tiered-ImageNet.

We adopt the Attack Success Rate (ASR) to evaluate the attack performance of UAPs on 2000 FSL tasks. The generator network, following the approach described in [52], was optimized using the Adam optimizer with an initial learning rate of 0.0002. The perturbation was kept within an $\ell_\infty$-norm bound of $\epsilon = 10$, considering pixel values in the range of 0 to 255. Additional FSL testing configurations follow the details provided in Libfewshot.

## 6.2 Results

**Effectiveness of our attacking framework.** Our investigation into the effectiveness of our attacking framework (FSAFW) across different FSL models on different tasks is demonstrated in 1 and 2. For each FSL paradigm, we targeted four models pre-trained on *mini*-ImageNet and TieredImageNet, utilizing ResNet12 and ResNet18 as backbone architectures. We report the mean Attack Success Rate (ASR, %, top-1) alongside 95% confidence intervals. For the meta-based methods ANIL and R2D2, we chose the victim models trained by 5-way 1-shot episodes denoted as ANIL-1 and R2D2-1 in the table (ANIL-5 and R2D2-5 are present in Section A). Notably, when the victim's dataset is consistent with the proxy dataset, attack performance generally improves. Our proposed attacking framework can achieve comparable ASR even when the proxy dataset is different from the victim one.

Table 5: Comparison of different contrastive losses on ASR.

| Tasks | Ours | InfoNCE | SimCLR | MSE | Cosine | SupCon | feature scatter |
|-------|------|---------|--------|-----|--------|--------|-----------------|
| 5-way 1-shot | $\mathbf{81.56}_{\pm \mathbf{0.29}}$ | $77.67_{\pm 0.31}$ | $77.26_{\pm 0.31}$ | $64.19_{\pm 0.33}$ | $79.32_{\pm 0.30}$ | $77.48_{\pm 0.31}$ | $75.85_{\pm 0.31}$ |
| 5-way 5-shot | $\mathbf{79.00}_{\pm \mathbf{0.18}}$ | $75.40_{\pm 0.23}$ | $74.69_{\pm 0.23}$ | $60.38_{\pm 0.29}$ | $76.87_{\pm 0.22}$ | $74.94_{\pm 0.23}$ | $72.90_{\pm 0.23}$ |

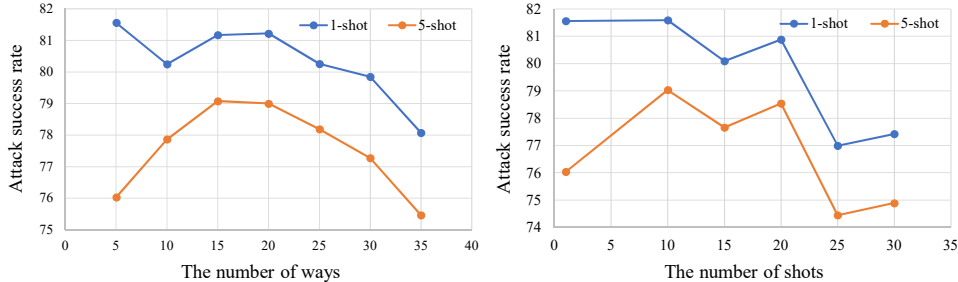

Figure 6: An illustration of the ASR that different forms of proxy tasks bring.

**Comparison with other attacking methods.** We compare the ASR of our attacking framework with the classic generator-based adversarial methods, such as UAN [15], GAP [30], and AdvEncoder [52], against the aforementioned FSL victim models. For this comparison, we utilized ResNet12 as the backbone and CIFAR-FS as the proxy dataset. ASR results for both 5-way 1-shot and 5-way 5-shot tasks are detailed in 3 and 4, respectively. Our framework consistently outperforms the other methods across victim models trained under various configurations. Note that some attacking methods rely on a fixed classifier in testing to generate UAPs, while models like R2D2, ProtoNet, and DN4 do not have one. These methods are not applicable to such models, and we have indicated this incompatibility with a dash ('-') in the relevant table cells.

### 6.3 Ablation Study

**The attack performance of contrastive losses.** To generalize the attack ability to the downstream tasks, a simple way is to adopt the contrastive losses[52]. In this section, we probe the effectiveness of various contrastive losses in crafting generalizable UAPs. We adopt the InfoNCE loss[29], SimCLR loss[6], MSE loss, Cosine similarity loss, SupCon loss[18] and feature scatter loss[51] to train the generator. We use a victim model trained on *mini*-ImageNet and with a backbone of ResNet12. The model is trained and tested under the Baseline[10] paradigm. We compare their ASR performance with that of our method in Table 5. The results suggest that contrastive losses in attacking FSL tasks are inferior to our FSAFW.

**The influence of different forms of proxy tasks.** We evaluate how different shapes of the proxy tasks affect the attack performance of the generated UAP. For the proxy tasks, we keep the number of shots constant at 1 while varying the number of ways, and vice versa, maintaining 5 ways while changing the number of shots. We utilize 5-way 5-shot and 5-way 1-shot as the shapes for downstream tasks. The attack results are displayed in the left and right panels of Figure 6. It can be observed that when the shapes of the proxy tasks do not deviate too much from the downstream tasks, the attack performance can be maintained. This suggests that when constructing the proxy tasks, a rough estimation of the downstream tasks' shape is enough.

## 7 Conclusion

In this work, we propose a unifying Few-Shot Attacking FrameWork (FSAFW) to generate transferable UAPs on the FSL scenario. We identify and analyze two major challenges in FSL: the task shift and the semantic shift. We adopt proxy tasks to handle the task shift and proxy prototypes to address the semantic shift. Our framework significantly outperforms existing methods, establishing a new benchmark for launching attacks on FSL tasks.

## Acknowledgements

This work is supported by the National Natural Science Foundation of China under grants 62206102, 62376103, 62436003 and 62302184; the Science and Technology Support Program of Hubei Province under grant 2022BAA046; Hubei Science and Technology Talent Service Project under grant 2024DJC078; and Ant Group through CCF-Ant Research Fund.

## References

[1] Yuanhao Ban and Yinpeng Dong. Pre-trained adversarial perturbations. *Advances in Neural Information Processing Systems*, 35:1196–1209, 2022.

[2] Peyman Bateni, Raghav Goyal, Vaden Masrani, Frank Wood, and Leonid Sigal. Improved few-shot visual classification. In *Proceedings of the IEEE/CVF Conference on Computer Vision and Pattern Recognition*, pages 14493–14502, 2020.

[3] Luca Bertinetto, Joao F Henriques, Philip HS Torr, and Andrea Vedaldi. Meta-learning with differentiable closed-form solvers. *arXiv preprint arXiv:1805.08136*, 2018.

[4] Luca Bertinetto, Joao F Henriques, Philip HS Torr, and Andrea Vedaldi. Meta-learning with differentiable closed-form solvers. *arXiv preprint arXiv:1805.08136*, 2018.

[5] Nicholas Carlini and David Wagner. Towards evaluating the robustness of neural networks. In *2017 ieee symposium on security and privacy (sp)*, pages 39–57. Ieee, 2017.

[6] Ting Chen, Simon Kornblith, Mohammad Norouzi, and Geoffrey Hinton. A simple framework for contrastive learning of visual representations. In *International conference on machine learning*, pages 1597–1607. PMLR, 2020.

[7] Wei-Yu Chen, Yen-Cheng Liu, Zsolt Kira, Yu-Chiang Frank Wang, and Jia-Bin Huang. A closer look at few-shot classification. *arXiv preprint arXiv:1904.04232*, 2019.

[8] Wei-Yu Chen, Yen-Cheng Liu, Zsolt Kira, Yu-Chiang Frank Wang, and Jia-Bin Huang. A closer look at few-shot classification. *arXiv preprint arXiv:1904.04232*, 2019.

[9] Jia Deng, Wei Dong, Richard Socher, Li-Jia Li, Kai Li, and Li Fei-Fei. Imagenet: A large-scale hierarchical image database. In *2009 IEEE Conference on Computer Vision and Pattern Recognition*, pages 248–255, 2009.

[10] Guneet S Dhillon, Pratik Chaudhari, Avinash Ravichandran, and Stefano Soatto. A baseline for few-shot image classification. *arXiv preprint arXiv:1909.02729*, 2019.

[11] Yinpeng Dong, Fangzhou Liao, Tianyu Pang, Hang Su, Jun Zhu, Xiaolin Hu, and Jianguo Li. Boosting adversarial attacks with momentum. In *Proceedings of the IEEE conference on computer vision and pattern recognition*, pages 9185–9193, 2018.

[12] Alexey Dosovitskiy, Lucas Beyer, Alexander Kolesnikov, Dirk Weissenborn, Xiaohua Zhai, Thomas Unterthiner, Mostafa Dehghani, Matthias Minderer, Georg Heigold, Sylvain Gelly, et al. An image is worth 16x16 words: Transformers for image recognition at scale. *arXiv preprint arXiv:2010.11929*, 2020.

[13] Chelsea Finn, Pieter Abbeel, and Sergey Levine. Model-agnostic meta-learning for fast adaptation of deep networks. In *International conference on machine learning*, pages 1126–1135. PMLR, 2017.

[14] Ian J Goodfellow, Jonathon Shlens, and Christian Szegedy. Explaining and harnessing adversarial examples. *arXiv preprint arXiv:1412.6572*, 2014.

[15] Jamie Hayes and George Danezis. Learning universal adversarial perturbations with generative models. In *2018 IEEE Security and Privacy Workshops (SPW)*, pages 43–49. IEEE, 2018.

[16] Kaiming He, Xiangyu Zhang, Shaoqing Ren, and Jian Sun. Deep residual learning for image recognition. In *Proceedings of the IEEE conference on computer vision and pattern recognition*, pages 770–778, 2016.

[17] Justin Johnson, Alexandre Alahi, and Li Fei-Fei. Perceptual losses for real-time style transfer and super-resolution. In *Computer Vision–ECCV 2016: 14th European Conference, Amsterdam, The Netherlands, October 11-14, 2016, Proceedings, Part II 14*, pages 694–711. Springer, 2016.

[18] Prannay Khosla, Piotr Teterwak, Chen Wang, Aaron Sarna, Yonglong Tian, Phillip Isola, Aaron Maschinot, Ce Liu, and Dilip Krishnan. Supervised contrastive learning. *Advances in neural information processing systems*, 33:18661–18673, 2020.

[19] Alex Krizhevsky, Geoffrey Hinton, et al. Learning multiple layers of features from tiny images. 2009.

[20] Wenbin Li, Lei Wang, Jinglin Xu, Jing Huo, Yang Gao, and Jiebo Luo. Revisiting local descriptor based image-to-class measure for few-shot learning. In *Proceedings of the IEEE/CVF conference on computer vision and pattern recognition*, pages 7260–7268, 2019.

[21] Wenbin Li, Ziyi Wang, Xuesong Yang, Chuanqi Dong, Pinzhuo Tian, Tiexin Qin, Jing Huo, Yinghuan Shi, Lei Wang, Yang Gao, et al. Libfewshot: A comprehensive library for few-shot learning. *IEEE Transactions on Pattern Analysis and Machine Intelligence*, 2023.

[22] Yanpei Liu, Xinyun Chen, Chang Liu, and Dawn Song. Delving into transferable adversarial examples and black-box attacks. *arXiv preprint arXiv:1611.02770*, 2016.

[23] Aleksander Madry, Aleksandar Makelov, Ludwig Schmidt, Dimitris Tsipras, and Adrian Vladu. Towards deep learning models resistant to adversarial attacks. *arXiv preprint arXiv:1706.06083*, 2017.

[24] Puneet Mangla, Nupur Kumari, Abhishek Sinha, Mayank Singh, Balaji Krishnamurthy, and Vineeth N Balasubramanian. Charting the right manifold: Manifold mixup for few-shot learning. In *Proceedings of the IEEE/CVF winter conference on applications of computer vision*, pages 2218–2227, 2020.

[25] Erik G Miller, Nicholas E Matsakis, and Paul A Viola. Learning from one example through shared densities on transforms. In *Proceedings IEEE Conference on Computer Vision and Pattern Recognition. CVPR 2000 (Cat. No. PR00662)*, volume 1, pages 464–471. IEEE, 2000.

[26] Seyed-Mohsen Moosavi-Dezfooli, Alhussein Fawzi, Omar Fawzi, and Pascal Frossard. Universal adversarial perturbations. In *Proceedings of the IEEE conference on computer vision and pattern recognition*, pages 1765–1773, 2017.

[27] Seyed-Mohsen Moosavi-Dezfooli, Alhussein Fawzi, and Pascal Frossard. Deepfool: a simple and accurate method to fool deep neural networks. In *Proceedings of the IEEE conference on computer vision and pattern recognition*, pages 2574–2582, 2016.

[28] Konda Reddy Mopuri, Utsav Garg, and R Venkatesh Babu. Fast feature fool: A data independent approach to universal adversarial perturbations. *arXiv preprint arXiv:1707.05572*, 2017.

[29] Aaron van den Oord, Yazhe Li, and Oriol Vinyals. Representation learning with contrastive predictive coding. *arXiv preprint arXiv:1807.03748*, 2018.

[30] Omid Poursaeed, Isay Katsman, Bicheng Gao, and Serge Belongie. Generative adversarial perturbations. In *Proceedings of the IEEE conference on computer vision and pattern recognition*, pages 4422–4431, 2018.

[31] Aniruddh Raghu, Maithra Raghu, Samy Bengio, and Oriol Vinyals. Rapid learning or feature reuse? towards understanding the effectiveness of maml. *arXiv preprint arXiv:1909.09157*, 2019.

[32] Jathushan Rajasegaran, Salman Khan, Munawar Hayat, Fahad Shahbaz Khan, and Mubarak Shah. Self-supervised knowledge distillation for few-shot learning. *arXiv preprint arXiv:2006.09785*, 2020.

[33] Sachin Ravi and Hugo Larochelle. Optimization as a model for few-shot learning. In *International conference on learning representations*, 2017.

[34] Mengye Ren, Eleni Triantafillou, Sachin Ravi, Jake Snell, Kevin Swersky, Joshua B Tenenbaum, Hugo Larochelle, and Richard S Zemel. Meta-learning for semi-supervised few-shot classification. *arXiv preprint arXiv:1803.00676*, 2018.

[35] Adam Santoro, Sergey Bartunov, Matthew Botvinick, Daan Wierstra, and Timothy Lillicrap. Meta-learning with memory-augmented neural networks. In *International conference on machine learning*, pages 1842–1850. PMLR, 2016.

[36] Jake Snell, Kevin Swersky, and Richard Zemel. Prototypical networks for few-shot learning. *Advances in neural information processing systems*, 30, 2017.

[37] Johannes Stallkamp, Marc Schlipsing, Jan Salmen, and Christian Igel. The german traffic sign recognition benchmark: a multi-class classification competition. In *The 2011 international joint conference on neural networks*, pages 1453–1460. IEEE, 2011.

[38] Qianru Sun, Yaoyao Liu, Zhaozheng Chen, Tat-Seng Chua, and Bernt Schiele. Meta-transfer learning through hard tasks. *IEEE Transactions on Pattern Analysis and Machine Intelligence*, 44(3):1443–1456, 2020.

[39] Qianru Sun, Yaoyao Liu, Tat-Seng Chua, and Bernt Schiele. Meta-transfer learning for few-shot learning. In *Proceedings of the IEEE/CVF conference on computer vision and pattern recognition*, pages 403–412, 2019.

[40] Flood Sung, Yongxin Yang, Li Zhang, Tao Xiang, Philip HS Torr, and Timothy M Hospedales. Learning to compare: Relation network for few-shot learning. In *Proceedings of the IEEE conference on computer vision and pattern recognition*, pages 1199–1208, 2018.

[41] Christian Szegedy, Wojciech Zaremba, Ilya Sutskever, Joan Bruna, Dumitru Erhan, Ian Goodfellow, and Rob Fergus. Intriguing properties of neural networks. *arXiv preprint arXiv:1312.6199*, 2013.

[42] Yonglong Tian, Yue Wang, Dilip Krishnan, Joshua B Tenenbaum, and Phillip Isola. Rethinking few-shot image classification: a good embedding is all you need? In *Computer Vision–ECCV 2020: 16th European Conference, Glasgow, UK, August 23–28, 2020, Proceedings, Part XIV 16*, pages 266–282. Springer, 2020.

[43] Oriol Vinyals, Charles Blundell, Timothy Lillicrap, Daan Wierstra, et al. Matching networks for one shot learning. *Advances in neural information processing systems*, 29, 2016.

[44] Oriol Vinyals, Charles Blundell, Timothy Lillicrap, Daan Wierstra, et al. Matching networks for one shot learning. *Advances in neural information processing systems*, 29, 2016.

[45] Catherine Wah, Steve Branson, Peter Welinder, Pietro Perona, and Serge Belongie. The caltech-ucsd birds-200-2011 dataset. 2011.

[46] Jing Xu, Xu Luo, Xinglin Pan, Yanan Li, Wenjie Pei, and Zenglin Xu. Alleviating the sample selection bias in few-shot learning by removing projection to the centroid. *Advances in Neural Information Processing Systems*, 35:21073–21086, 2022.

[47] Shuo Yang, Lu Liu, and Min Xu. Free lunch for few-shot learning: Distribution calibration. *arXiv preprint arXiv:2101.06395*, 2021.

[48] Sung Whan Yoon, Jun Seo, and Jaekyun Moon. Tapnet: Neural network augmented with task-adaptive projection for few-shot learning. In *International conference on machine learning*, pages 7115–7123. PMLR, 2019.

[49] Chaoning Zhang, Philipp Benz, Tooba Imtiaz, and In So Kweon. Understanding adversarial examples from the mutual influence of images and perturbations. In *Proceedings of the IEEE/CVF Conference on Computer Vision and Pattern Recognition*, pages 14521–14530, 2020.

[50] Chi Zhang, Yujun Cai, Guosheng Lin, and Chunhua Shen. Deepemd: Few-shot image classification with differentiable earth mover's distance and structured classifiers. In *Proceedings of the IEEE/CVF conference on computer vision and pattern recognition*, pages 12203–12213, 2020.

[51] Haichao Zhang and Jianyu Wang. Defense against adversarial attacks using feature scattering-based adversarial training. *Advances in Neural Information Processing Systems*, 32, 2019.

[52] Ziqi Zhou, Shengshan Hu, Ruizhi Zhao, Qian Wang, Leo Yu Zhang, Junhui Hou, and Hai Jin. Downstream-agnostic adversarial examples. In *Proceedings of the IEEE/CVF International Conference on Computer Vision*, pages 4345–4355, 2023.

Table 6: 5-way 1-shot ASR results of our framework in different FSL victim models.

| Victim | Proxy | Backbone | ANIL-5 | R2D2-5 | ProtoNet-5 | DN4-5 |
|---|---|---|---|---|---|---|
| Mini | Cifar | RN12 | $67.80_{\pm 0.45}$ | $74.77_{\pm 0.34}$ | $61.10_{\pm 0.39}$ | $66.70_{\pm 0.37}$ |
| | | RN18 | $59.42_{\pm 0.45}$ | $65.95_{\pm 0.32}$ | $70.11_{\pm 0.34}$ | $63.99_{\pm 0.32}$ |
| | Mini | RN12 | $69.48_{\pm 0.45}$ | $74.76_{\pm 0.34}$ | $68.80_{\pm 0.41}$ | $68.77_{\pm 0.35}$ |
| | | RN18 | $60.14_{\pm 0.45}$ | $79.12_{\pm 0.28}$ | $70.45_{\pm 0.35}$ | $65.36_{\pm 0.32}$ |
| | Tiered | RN12 | $69.31_{\pm 0.43}$ | $74.66_{\pm 0.35}$ | $68.34_{\pm 0.41}$ | $70.22_{\pm 0.35}$ |
| | | RN18 | $60.06_{\pm 0.47}$ | $78.68_{\pm 0.28}$ | $65.11_{\pm 0.32}$ | $65.32_{\pm 0.32}$ |

Table 7: 5-way 5-shot ASR results of our framework in different FSL victim models.

| Victim | Proxy | Backbone | ANIL-5 | R2D2-5 | ProtoNet-5 | DN4-5 |
|---|---|---|---|---|---|---|
| Mini | Cifar | RN12 | $72.75_{\pm 0.28}$ | $79.39_{\pm 0.17}$ | $63.65_{\pm 0.31}$ | $69.05_{\pm 0.26}$ |
| | | RN18 | $67.93_{\pm 0.26}$ | $65.90_{\pm 0.25}$ | $76.06_{\pm 0.18}$ | $64.32_{\pm 0.27}$ |
| | Mini | RN12 | $74.96_{\pm 0.25}$ | $79.40_{\pm 0.17}$ | $75.84_{\pm 0.20}$ | $74.30_{\pm 0.23}$ |
| | | RN18 | $71.93_{\pm 0.25}$ | $79.48_{\pm 0.16}$ | $75.53_{\pm 0.18}$ | $66.28_{\pm 0.27}$ |
| | Tiered | RN12 | $74.51_{\pm 0.26}$ | $79.38_{\pm 0.17}$ | $75.80_{\pm 0.19}$ | $74.44_{\pm 0.23}$ |
| | | RN18 | $69.07_{\pm 0.26}$ | $79.33_{\pm 0.16}$ | $68.52_{\pm 0.24}$ | $66.18_{\pm 0.27}$ |

# A    More Victim Models

Due to page limitations, we present the Attack Success Rate (ASR) of our method on more victim models trained under different paradigms in this section. Specifically, we selected the victim models trained by 5-way 5-shot episodes under the four paradigms: ANIL [31], R2D2 [3], ProtoNet [36], and DN4 [20]. These models are referred to as ANIL-5, R2D2-5, ProtoNet-5, and DN4-5 in the table. The victim dataset we chose is *mini*-ImageNet, while CIFAR-FS, *mini*-ImageNet, and tieredImageNet were used as proxy datasets. The corresponding results can be found in Table 6 and Table 7. As observed, our method consistently maintains a high Attack Success Rate (ASR) across all victim models, across different Few-Shot Learning (FSL) settings. This demonstrates the effectiveness of our method on models trained with 5-way 5-shot episodes.

# B    Compared with An Additional UAP Method

We have incorporated the PAP[1] method under the few-shot learning scenario and made a comparative analysis. The results in Table 8 show that our proposed method demonstrates effectiveness not only on the base dataset but also exhibits superior cross-dataset performance without necessitating access to the base dataset.

Table 8: The ASR results compared with the PAP method.

| Proxy | Method | 1-shot | | | 5-shot | | |
|---|---|---|---|---|---|---|---|
| | | Baseline | ANIL-1 | ProtoNet-1 | Baseline | ANIL-1 | ProtoNet-1 |
| Cifar | PAP | $44.39_{\pm 0.30}$ | $44.52_{\pm 0.29}$ | $37.70_{\pm 0.29}$ | $36.58_{\pm 0.26}$ | $42.34_{\pm 0.28}$ | $35.69_{\pm 0.26}$ |
| | Ours | $\mathbf{81.56_{\pm 0.29}}$ | $\mathbf{77.84_{\pm 0.28}}$ | $\mathbf{69.03_{\pm 0.36}}$ | $\mathbf{79.00_{\pm 0.18}}$ | $\mathbf{78.31_{\pm 0.21}}$ | $\mathbf{67.96_{\pm 0.28}}$ |
| CUB | PAP | $45.59_{\pm 0.30}$ | $44.73_{\pm 0.29}$ | $37.92_{\pm 0.30}$ | $37.86_{\pm 0.26}$ | $42.65_{\pm 0.27}$ | $36.21_{\pm 0.26}$ |
| | Ours | $\mathbf{81.17_{\pm 0.30}}$ | $\mathbf{77.83_{\pm 0.28}}$ | $\mathbf{71.22_{\pm 0.36}}$ | $\mathbf{78.29_{\pm 0.21}}$ | $\mathbf{78.25_{\pm 0.20}}$ | $\mathbf{72.24_{\pm 0.28}}$ |
| Omniglot | PAP | $47.91_{\pm 0.31}$ | $45.88_{\pm 0.28}$ | $38.52_{\pm 0.29}$ | $40.69_{\pm 0.27}$ | $43.92_{\pm 0.27}$ | $36.63_{\pm 0.26}$ |
| | Ours | $\mathbf{76.32_{\pm 0.31}}$ | $\mathbf{77.81_{\pm 0.28}}$ | $\mathbf{67.70_{\pm 0.35}}$ | $\mathbf{74.46_{\pm 0.21}}$ | $\mathbf{77.87_{\pm 0.21}}$ | $\mathbf{69.05_{\pm 0.26}}$ |
| GTSRB | PAP | $42.62_{\pm 0.29}$ | $44.54_{\pm 0.29}$ | $37.96_{\pm 0.29}$ | $34.67_{\pm 0.26}$ | $42.39_{\pm 0.27}$ | $49.44_{\pm 0.26}$ |
| | Ours | $\mathbf{79.99_{\pm 0.30}}$ | $\mathbf{78.13_{\pm 0.20}}$ | $\mathbf{69.17_{\pm 0.38}}$ | $\mathbf{77.55_{\pm 0.20}}$ | $\mathbf{77.93_{\pm 0.28}}$ | $\mathbf{69.98_{\pm 0.30}}$ |

Table 9: 5-way 1-shot ASR results of cross domain datasets.

| Proxy | Downstream | Method | Baseline | ANIL-1 | ProtoNet-1 |
|---|---|---|---|---|---|
| CUB | Mini | UAN | $52.47_{\pm 0.31}$ | $62.85_{\pm 0.27}$ | - |
| | | GAP | $68.47_{\pm 0.31}$ | $67.47_{\pm 0.28}$ | - |
| | | AdvEncoder | $76.57_{\pm 0.31}$ | $72.86_{\pm 0.27}$ | $70.80_{\pm 0.36}$ |
| | | Ours | $\mathbf{81.17_{\pm 0.30}}$ | $\mathbf{77.83_{\pm 0.28}}$ | $\mathbf{71.22_{\pm 0.36}}$ |
| Omniglot | Mini | UAN | $44.90_{\pm 0.30}$ | $37.58_{\pm 0.26}$ | - |
| | | GAP | $60.95_{\pm 0.33}$ | $63.82_{\pm 0.35}$ | - |
| | | AdvEncoder | $74.16_{\pm 0.33}$ | $47.97_{\pm 0.34}$ | $54.72_{\pm 0.31}$ |
| | | Ours | $\mathbf{76.32_{\pm 0.31}}$ | $\mathbf{77.81_{\pm 0.28}}$ | $\mathbf{67.70_{\pm 0.35}}$ |
| GTSRB | Mini | UAN | $54.45_{\pm 0.32}$ | $50.17_{\pm 0.26}$ | - |
| | | GAP | $54.09_{\pm 0.33}$ | $68.22_{\pm 0.31}$ | - |
| | | AdvEncoder | $67.24_{\pm 0.32}$ | $62.86_{\pm 0.30}$ | $67.16_{\pm 0.34}$ |
| | | Ours | $\mathbf{79.99_{\pm 0.30}}$ | $\mathbf{78.13_{\pm 0.20}}$ | $\mathbf{69.17_{\pm 0.38}}$ |
| Mini | CUB | UAN | $76.82_{\pm 0.36}$ | $73.75_{\pm 0.48}$ | - |
| | | GAP | $69.70_{\pm 0.35}$ | $75.08_{\pm 0.45}$ | - |
| | | AdvEncoder | $76.21_{\pm 0.35}$ | $76.82_{\pm 0.56}$ | $71.40_{\pm 0.40}$ |
| | | Ours | $\mathbf{79.22_{\pm 0.35}}$ | $\mathbf{78.09_{\pm 0.58}}$ | $\mathbf{71.60_{\pm 0.40}}$ |
| Mini | Omniglot | UAN | $36.20_{\pm 0.40}$ | $56.63_{\pm 0.59}$ | - |
| | | GAP | $20.09_{\pm 0.32}$ | $25.40_{\pm 0.33}$ | - |
| | | AdvEncoder | $64.41_{\pm 0.41}$ | $66.27_{\pm 0.65}$ | $71.80_{\pm 0.38}$ |
| | | Ours | $\mathbf{76.44_{\pm 0.27}}$ | $\mathbf{69.64_{\pm 0.64}}$ | $\mathbf{75.27_{\pm 0.36}}$ |
| Mini | GTSRB | UAN | $74.86_{\pm 0.34}$ | $79.82_{\pm 0.33}$ | - |
| | | GAP | $73.30_{\pm 0.34}$ | $79.04_{\pm 0.36}$ | - |
| | | AdvEncoder | $77.18_{\pm 0.33}$ | $78.27_{\pm 0.37}$ | $76.71_{\pm 0.37}$ |
| | | Ours | $\mathbf{80.29_{\pm 0.30}}$ | $\mathbf{80.31_{\pm 0.36}}$ | $\mathbf{78.41_{\pm 0.35}}$ |

## C   Cross Domain Experiments

To better validate the generalizability of our attack approach, we conducted experiments on various cross-domain datasets. We used CUB [45], [44], and GTSRB [37] as cross-domain datasets, employing them both as proxy datasets and downstream datasets to implement the attacks. We selected one victim model each from the fine-tuning, meta-based, and metric-based FSL paradigms for our experiments, comparing our approach with previous attack methods. The results are shown in Table 9 and Table 10, demonstrating that our attacking framework exhibits broader generalizability.

## D   Datasets

The CIFAR-FS [4] dataset is randomly sampled from CIFAR-100 [19] which contains 100 classes with 600 images of 32×32 size per class. The 100 classes are split into 64 classes for training, 16 classes for validation and 20 classes for testing. The average inter-class similarity is sufficiently high to represent a challenge for the current state of the art. Moreover, the limited original resolution of 32×32 makes the task harder and at the same time allows fast prototyping. The samples of this dataset are listed in Figure 7.

The *mini*-ImageNet dataset was proposed by [43] for few-shot learning evaluation. Its complexity is high due to the use of ImageNet images but requires fewer resources and infrastructure than running on the full ImageNet dataset. In total, there are 100 classes with 600 samples of 84×84 color images per class. These 100 classes are divided into 64, 16, and 20 classes respectively for sampling tasks for meta-training, meta-validation, and meta-test. The samples of this dataset are listed in Figure 8

The tieredImageNet dataset [34] is a larger subset of ILSVRC-12 with 608 classes (779,165 images) grouped into 34 higher-level nodes in the ImageNet human-curated hierarchy. This set of nodes is partitioned into 20, 6, and 8 disjoint sets of training, validation, and testing nodes, and the corresponding classes form the respective meta-sets. As argued in Ren et al., this split near the root

Table 10: 5-way 5-shot ASR results of cross domain datasets.

| Proxy | Downstream | Method | Baseline | ANIL-1 | ProtoNet-1 |
|---|---|---|---|---|---|
| CUB | Mini | UAN | $45.93_{\pm0.28}$ | $63.53_{\pm0.24}$ | - |
| | | GAP | $65.91_{\pm0.24}$ | $67.58_{\pm0.25}$ | - |
| | | AdvEncoder | $73.67_{\pm0.23}$ | $72.67_{\pm0.22}$ | $71.99_{\pm0.26}$ |
| | | Ours | $\mathbf{78.29_{\pm0.21}}$ | $\mathbf{78.25_{\pm0.20}}$ | $\mathbf{72.24_{\pm0.28}}$ |
| Omniglot | Mini | UAN | $37.34_{\pm0.27}$ | $35.05_{\pm0.26}$ | - |
| | | GAP | $57.22_{\pm0.29}$ | $61.09_{\pm0.31}$ | - |
| | | AdvEncoder | $72.07_{\pm0.23}$ | $44.73_{\pm0.31}$ | $57.38_{\pm0.26}$ |
| | | Ours | $\mathbf{74.46_{\pm0.21}}$ | $\mathbf{77.87_{\pm0.21}}$ | $\mathbf{69.05_{\pm0.26}}$ |
| GTSRB | Mini | UAN | $48.46_{\pm0.29}$ | $48.73_{\pm0.26}$ | - |
| | | GAP | $48.68_{\pm0.29}$ | $65.94_{\pm0.28}$ | - |
| | | AdvEncoder | $63.99_{\pm0.27}$ | $60.51_{\pm0.27}$ | $68.96_{\pm0.24}$ |
| | | Ours | $\mathbf{77.55_{\pm0.20}}$ | $\mathbf{77.93_{\pm0.28}}$ | $\mathbf{69.98_{\pm0.30}}$ |
| Mini | CUB | UAN | $77.42_{\pm0.20}$ | $64.42_{\pm0.58}$ | - |
| | | GAP | $68.90_{\pm0.27}$ | $68.89_{\pm0.52}$ | - |
| | | AdvEncoder | $75.29_{\pm0.22}$ | $67.31_{\pm0.68}$ | $72.80_{\pm0.27}$ |
| | | Ours | $\mathbf{77.46_{\pm0.21}}$ | $\mathbf{69.43_{\pm0.71}}$ | $\mathbf{72.86_{\pm0.27}}$ |
| Mini | Omniglot | UAN | $26.15_{\pm0.39}$ | $52.25_{\pm0.60}$ | - |
| | | GAP | $10.14_{\pm0.23}$ | $21.69_{\pm0.32}$ | - |
| | | AdvEncoder | $60.57_{\pm0.44}$ | $62.99_{\pm0.66}$ | $68.25_{\pm0.34}$ |
| | | Ours | $\mathbf{75.04_{\pm0.25}}$ | $\mathbf{67.52_{\pm0.67}}$ | $\mathbf{73.15_{\pm0.30}}$ |
| Mini | GTSRB | UAN | $74.27_{\pm0.23}$ | $78.31_{\pm0.28}$ | - |
| | | GAP | $68.57_{\pm0.29}$ | $77.77_{\pm0.29}$ | - |
| | | AdvEncoder | $73.96_{\pm0.26}$ | $77.50_{\pm0.29}$ | $78.38_{\pm0.23}$ |
| | | Ours | $\mathbf{77.22_{\pm0.24}}$ | $\mathbf{78.50_{\pm0.28}}$ | $\mathbf{78.77_{\pm0.23}}$ |

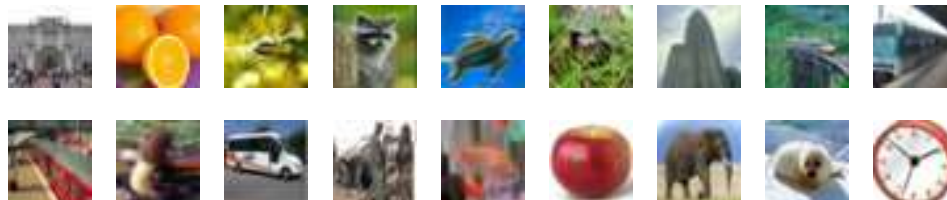

Figure 7: Samples of CIFAR-FS.

of the ImageNet hierarchy results in a more challenging, yet realistic regime with test classes that are less similar to training classes. The samples of this dataset are listed in Figure 9

## E   Broader Impact

We point out the security problem in a widely used open-set setting, i.e. Few-Shot Learning (FSL). Under this setting, we propose a unified attack framework to generate the UAP, which can fool all kinds of FSL models. The attack method can be used in many vision-based real-world scenarios, as

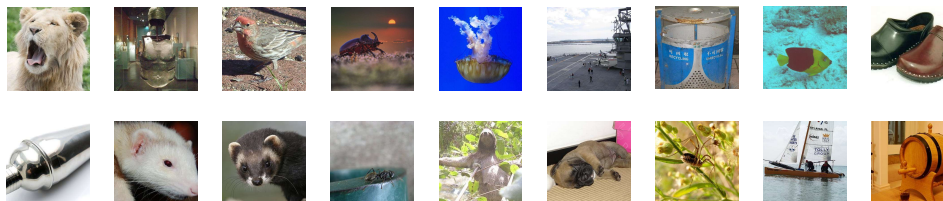

Figure 8: Samples of *mini*-ImageNet.

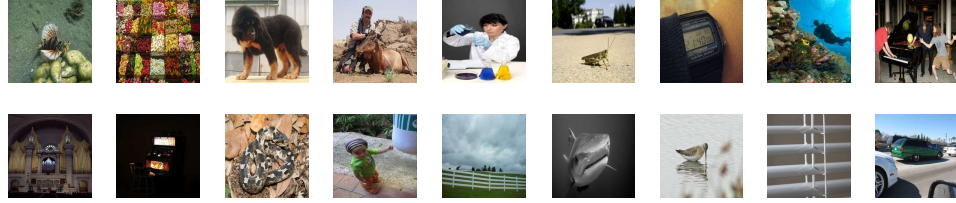

Figure 9: Samples of tieredImageNet.

it doesn't need to acquire the pre-training and downstream data. Also, the attack method can draw attention to the development of a defense method in FSL.

